# Sparse Representation and Its Applications in Blind Source Separation

**Yuanqing Li, Andrzej Cichocki, Shun-ichi Amari, Sergei Shishkin**
RIKEN Brain Science Institute, Saitama, 3510198, Japan

**Jianting Cao**
Department of Electronic Engineering
Saitama Institute of Technology
Saitama, 3510198, Japan

**Fanji Gu**
Department of Physiology and Biophysics
Fudan University
Shanghai, China

## Abstract

In this paper, sparse representation (factorization) of a data matrix is first discussed. An overcomplete basis matrix is estimated by using the $K-$means method. We have proved that for the estimated overcomplete basis matrix, the sparse solution (coefficient matrix) with minimum $l^1-$norm is unique with probability of one, which can be obtained using a linear programming algorithm. The comparisons of the $l^1-$norm solution and the $l^0-$norm solution are also presented, which can be used in recoverability analysis of blind source separation (BSS). Next, we apply the sparse matrix factorization approach to BSS in the overcomplete case. Generally, if the sources are not sufficiently sparse, we perform blind separation in the time-frequency domain after preprocessing the observed data using the wavelet packets transformation. Third, an EEG experimental data analysis example is presented to illustrate the usefulness of the proposed approach and demonstrate its performance. Two almost independent components obtained by the sparse representation method are selected for phase synchronization analysis, and their periods of significant phase synchronization are found which are related to tasks. Finally, concluding remarks review the approach and state areas that require further study.

## 1   Introduction

Sparse representation or sparse coding of signals has received a great deal of attention in recent years. For instance, sparse representation of signals using large-scale linear programming under given overcomplete bases (e.g., wavelets) was discussed in [1]. Also, in [2], a sparse image coding approach using the wavelet pyramid architecture was presented. Sparse representation can be used in blind source separation [3][4]. In [3], a two stage approach was proposed, that is, the first is to estimate the mixing matrix by using a clustering algorithm, the second is to estimate the source matrix. In our opinion, there are still three fundamental problems related to sparse representation of signals and BSS which need to be

further studied: 1) detailed recoverability analysis; 2) high dimensionality of the observed data; 3) overcomplete case in which the sources number is unknown.

The present paper first considers sparse representation (factorization) of a data matrix based on the following model

$$\mathbf{X} = \mathbf{BS}, \tag{1}$$

where the $\mathbf{X} = [\mathbf{x}(1), \cdots, \mathbf{x}(N)] \in R^{n \times N}$ ($N \gg 1$) is a known data matrix, $\mathbf{B} = [\mathbf{b}_1 \cdots \mathbf{b}_m]$ is a $n \times m$ basis matrix, $\mathbf{S} = [\mathbf{s}_1, \cdots, \mathbf{s}_N] = [s_{ij}]_{m \times N}$ is a coefficient matrix, also called a solution corresponding to the basis matrix $\mathbf{B}$. Generally, $m > n$, which implies that the basis is overcomplete.

The discussion of this paper is under the following assumptions on (1).

**Assumption 1:** 1. The number of basis vectors $m$ is assumed to be fixed in advance and satisfies the condition $n \le m < N$. 2. All basis vectors are normalized to be unit vectors with their $2-$norms being equal to $1$ and all $n$ basis vectors are linearly independent.

The rest of this paper is organized as follows. Section 2 analyzes the sparse representation of a data matrix. Section 3 presents the comparison of the $l^0$ norm solution and $l^1$ norm solution. Section 4 discusses blind source separation via sparse representation. An EEG data analysis example is given in Section 5. Concluding remarks in Section 6 summarize the advantages of the proposed approach.

## 2   Sparse representation of data matrix

In this section, we discuss sparse representation of the data matrix $\mathbf{X}$ using the two-stage approach proposed in [3]. At first, we apply an algorithm based on $K-$means clustering method for finding a suboptimal basis matrix that is composed of the cluster centers of the normalized, known data vectors as in [3]. With this kind of cluster center basis matrix, the corresponding coefficient matrix estimated by linear programming algorithm presented in this section can become very sparse.

**Algorithm outline 1:** *Step 1.* Normalize the data vectors. *Step 2.* Begin a $K-$means clustering iteration followed by normalization to estimate the suboptimal basis matrix. End

Now we discuss the estimation of the coefficient matrix. For a given basis matrix $\mathbf{B}$ in (1), the coefficient matrix can be found by solving the following optimization problem as in many existing references (e.g., [3, 5]),

$$\min \sum_{i=1}^{m} \sum_{j=1}^{N} |s_{ij}|, \text{ subject to } \mathbf{BS} = \mathbf{X}. \tag{2}$$

It is not difficult to prove that the linear programming problem (2) is equivalent to the following set of $N$ smaller scale linear programming problems:

$$\min \sum_{i=1}^{m} |s_{ij}|, \text{ subject to } \mathbf{Bs}_j = \mathbf{x}(j), \ j = 1, \cdots, N. \tag{3}$$

By setting $\mathbf{S} = \mathbf{U} - \mathbf{V}$, where $\mathbf{U} = [u_{ij}]_{m \times N} \ge 0$, $\mathbf{V} = [v_{ij}]_{m \times N} \ge 0$, (3) can be converted to the following standard linear programming problems with non-negative constraints,

$$\min \sum_{i=1}^{m} (u_{ij} + v_{ij}), \text{ subject to } [\mathbf{B}, -\mathbf{B}][\mathbf{u}_j^T, \mathbf{v}_j^T]^T = \mathbf{x}(j), \ \mathbf{u}_j \ge 0, \ \mathbf{v}_j \ge 0, \tag{4}$$

where $j = 1, \cdots, N$.

**Theorem 1** *For almost all bases* $\mathbf{B} \in R^{n \times m}$*, the sparse solution* ($l^1-$*norm solution) of (1) is unique. That is, the set of bases* $\mathbf{B}$*, under which the sparse solution of (1) is not unique, is of measure zero. And there are at most* $n$ *nonzero entries of the solution.*

It follows from Theorem 1 that for any given basis, there exists a unique sparse solution of (2) with probability of one.

## 3   Comparison of the $l^0$ norm solution and $l^1$ norm solution

Usually, $l^0$ norm $J_0(\mathbf{S}) = \sum\limits_{i=1}^{n} \sum\limits_{j=1}^{N} |s_{ij}|^0$ (the number of nonzero entries of $\mathbf{S}$) is used as a sparsity measure of $\mathbf{S}$, since it ensures the sparsest solution. Under this measure, the sparse solution is obtained by solving the problem

$$\min \sum_{i=1}^{m} \sum_{j=1}^{N} |s_{ij}|^0, \text{ subject to } \mathbf{BS} = \mathbf{X}. \tag{5}$$

In [5], is discussed optimally sparse representation in general (non-orthogonal) dictionaries via $l^1-$norm minimization, and two sufficient conditions are proposed on the nonzero entry number of the $l^0-$norm solution, under which the equivalence between $l^0-$norm solution and $l^1-$norm solution holds precisely. However, these bounds are very small in real world situations generally, if the basis vectors are far away from orthogonality. For instance, the bound is smaller than $1.5$ in the simulation experiments shown in the next section. This implies that the $l^0-$norm solution allows only one nonzero entry in order that the equivalence holds. In the next, we will also discuss the equivalence of the $l^0$ norm solution and $l^1$ norm solution but from the viewpoint of probability.

First, we introduce the two optimization problems:

$$(P_0) \quad \min \sum_{i=1}^{m} |s_i|^0, \; subject \; to \; \mathbf{As} = \mathbf{x},$$

$$(P_1) \quad \min \sum_{i=1}^{m} |s_i|, \; subject \; to \; \mathbf{As} = \mathbf{x}.$$

where $\mathbf{A} \in R^{n \times m}$, $\mathbf{x} \in R^n$ are a known basis matrix and a data vector, respectively, and $\mathbf{s} \in R^m$, $n \leq m$. Suppose that $\mathbf{s}_{0*}$ is a solution of $(P_0)$, and $\mathbf{s}_{1*}$ is a solution of $(P_1)$.

**Theorem 2** *The solution of* $(P_0)$ *is not robust to additive noise of the model, while the solution of* $(P_1)$ *is robust to additive noise, at least to some degree.*

Although the problem $(P_0)$ provides the sparsest solution, it is not an efficient way to find the solution by solving the problem $(P_0)$. The reasons are: 1) if $||\mathbf{s}_{0*}||_0 = n$, then the solution of $(P_0)$ is not unique generally; 2) until now, an effective algorithm to solve the optimization problem $(P_0)$ does not exist (it has been proved that problem $(P_0)$ is NP hard); 3) the solution of $(P_0)$ is not robust to noise. In contrast, the solution of $(P_1)$ is unique with a probability of one according to Theorem 1. It is well known that there are many efficient optimization tools to solve the problem $(P_1)$. From the above mentioned facts arises naturally a problem: what is the condition under which the solution of $(P_1)$ is one of the sparsest solutions, that is, the solution has the same number of nonzero entries as the solution of $(P_0)$? In the following, we will discuss the problem.

**Lemma 1** *Suppose that* $\mathbf{x} \in R^n$ *and* $\mathbf{A} \in R^{n \times m}$ *are selected randomly. If* $\mathbf{x}$ *is represented by a linear combination of* $k$ *column vectors of* $\mathbf{A}$*, then* $k \geq n$ *generally, that is, the probability that* $k < n$ *is zero.*

**Theorem 3** *For the optimization problems $(P_0)$ and $(P_1)$, suppose that $\mathbf{A} \in R^{n \times m}$ is selected randomly, $\mathbf{x} \in R^n$ is generated by $\mathbf{A}\mathbf{s}_*$, $l = \|\mathbf{s}_*\|_0 < n$, and that all nonzero entries of $\mathbf{s}_*$ are also selected randomly. We have*

*1. $\mathbf{s}_*$ is the unique solution of $(P_0)$ with probability of one, that is, $\mathbf{s}_{0*} = \mathbf{s}_*$. And if $\|\mathbf{s}_{1*}\|_0 < n$, then $\mathbf{s}_{1*} = \mathbf{s}_*$ with probability of one. 2. The probability $P(\mathbf{s}_{1*} = \mathbf{s}_*) \geq (P(1, l, n, m))^l$, where $P(1, l, n, m)$ $(1 \leq l \leq n)$ are $n$ probabilities satisfying $1 = P(1, 1, n, m) \geq P(1, 2, n, m) \geq \cdots \geq P(1, n, n, m)$ (their explanations are omitted here due to limit of space). 3. For given positive integers $l_0$ and $n_0$, if $l \leq l_0$, and $m - n \leq n_0$, then $\lim\limits_{n \to +\infty} P(\mathbf{s}_{1*} = \mathbf{s}_*) = 1$.*

**Remarks 1:** 1. From Theorem 3, if $n$ and $m$ are fixed, and $l$ is sufficiently small, then $\mathbf{s}_{1*} = \mathbf{s}_*$ with a high probability. 2. For fixed $l$ and $m - n$, if $n$ is sufficiently large, then $\mathbf{s}_{1*} = \mathbf{s}_*$ with a high probability. Theorem 3 will be used in recoverability analysis of BSS.

## 4 Blind source separation based on sparse representation

In this section, we discuss blind source separation based on sparse representation of mixture signals. The proposed approach is also suitable for the case in which the number of sensors is less than or equal to the number of sources, while the number of source is unknown. We consider the following noise-free model,

$$\mathbf{x}_i = \mathbf{A}\mathbf{s}_i, \ i = 1, \cdots, N, \tag{6}$$

where the mixing matrix $\mathbf{A} \in R^{n \times m}$ is unknown, the matrix $\mathbf{S} = [\mathbf{s}_1, \cdots, \mathbf{s}_N] \in R^{m \times N}$ is composed by the $m$ unknown sources, and the only observed data matrix $\mathbf{X} = [\mathbf{x}_1, \cdots, \mathbf{x}_N] \in R^{n \times N}$ that has rows containing mixtures of sources, $n \leq m$. The task of blind source separation is to recover the sources using only the observable data matrix $\mathbf{X}$.

We also use a two-step approach presented in [3] for BSS. The first step is to estimate the mixing matrix using clustering Algorithm 1. If the mixing matrix is estimated correctly, and a source vector $\mathbf{s}_*$ satisfies that $\|\mathbf{s}_*\|_0 = l < n$, then by Theorem 3, $\mathbf{s}_*$ is the $l^0$-norm solution of (6) with probability one. And if the source vector is sufficiently sparse, e.g., $l$ is sufficiently small compared with $n$, then it can be recovered by solving the linear programming problem $(P_1)$ with a high probability. Considering the source number is unknown generally, we denote the estimated mixing matrix $\bar{\mathbf{A}} = [\tilde{\mathbf{A}}, \triangle\mathbf{A}] \in R^{n \times m'}$ $(m' > m)$. We introduce the following optimization problem $(P'_1)$ and denote its solution $\bar{\mathbf{s}} = [\tilde{\mathbf{s}}^T, \triangle\mathbf{s}^T]^T \in R^{m'}$,

$$(P'_1) \quad \min \sum_{i=1}^{m'} |s_i|, \ subject \ to \ \bar{\mathbf{A}}\mathbf{s} = \mathbf{x}.$$

We can prove the following recoverability result.

**Theorem 4** *Suppose that the sub-matrix $\tilde{\mathbf{A}}$ (of the estimated mixing matrix $\bar{\mathbf{A}}$) is sufficiently close to the true mixing matrix $\mathbf{A}$ neglecting scaling and permutation ambiguities, and that a source vector is sufficiently sparse. Then the source vector can be recovered with a high probability (close to one) by solving $(P'_1)$. That is, $\tilde{\mathbf{s}}$ is sufficiently close to the original source vector, and $\triangle\mathbf{s}$ is close to zero vector.*

To illustrate Theorem 4 partially, we have performed two simulation experiments in which the mixing matrix is supposed to be estimated correctly. Fig. 1 shows the probabilities that a source vector can be recovered correctly in different cases, estimated in the two

simulations. In the first simulation, $n$ and $m$ are fixed to be 10 and 15, respectively, $l$ denotes the number of nonzero entries of source vector and changes from 1 to 15. For every fixed nonzero entry number $l$, the probabilities that the source vector is recovered correctly is estimated through 3000 independent repeated stochastic experiments, in which the mixing matrix $\mathbf{A}$ and all nonzero entries of the source vector $\mathbf{s}_0$ are selected randomly according to the uniform distribution. Fig. 1 (a) shows the probability curve. We can see that the source can be estimated correctly when $l = 1, 2$, and the probability is greater than 0.95 when $l \leq 5$.

In the second simulation experiment, all original source vectors have 5 nonzero entries, that is, $l = 5$; and $m = 15$. The dimension $n$ of the mixture vectors varies from 5 to 15. As in the first simulation, the probabilities for correctly estimated source vectors are estimated through 3000 stochastic experiments and showed in Fig. 1 (b). It is evident that when $n \geq 10$, the source can be estimated correctly with probability higher than 0.95.

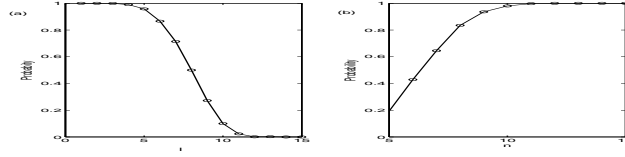

Figure 1: (a) the probability curve that the source vectors are estimated correctly as a function of $l$ obtained in the first simulation; (b) the probability curve that the source vectors are estimated correctly as a function of $n$ obtained in the second simulation.

In order to estimate the mixing matrix correctly, the sources should be sufficiently sparse. Thus sparseness of the sources plays an important role not only in estimating the sources but also in estimating the mixing matrix. However, if the sources are not sufficiently sparse in reality, we can have a wavelet packets transformation preprocessing. In the following, a blind separation algorithm based on preprocessing is presented for dense sources.

**Algorithm outline 2:**

*Step 1.* Transform the $n$ time domain signals, ($n$ rows of $\mathbf{X}$, to time-frequency signals by a wavelet packets transformation, and make sure that $n$ wavelet packets trees have the same structure.

*Step 2.* Select these nodes of wavelet packets trees, of which the coefficients are as sparse as possible. The selected nodes of different trees should have the same indices. Based on these coefficient vectors, estimate the mixing matrix $\bar{\mathbf{A}} \in R^{n \times m'}$ using the Algorithm 1 presented in Section 2.

*Step 3.* Based on the estimated mixing matrix $\bar{\mathbf{A}}$ and the coefficients of all nodes obtained in step 1, estimate the coefficients of all the nodes of the wavelet packets trees of sources by solving the set of linear programming problems (4).

*Step 4.* Reconstruct sources using the inverse wavelet packets transformation. End

We have successfully separated speech sources in a number of simulations in overcomplete case (e.g., 8 sources, 4 sensors) using Algorithm 2. In the next section, we will present an EEG data analysis example.

**Remark 2:** A challenge problem in the algorithm above is to estimate the mixing matrix as precisely as possible. In our many simulations on BSS of speech mixtures, we use $7-$level wavelet packets transformation for preprocessing. When $K-$means clustering method is used for estimating the mixing matrix, the number of clusters (the number of columns of the estimated mixing matrix) should be set to be greater than the source number even if the

source number is known. In this way, the estimated matrix will contain a submatrix very close to the original mixing matrix. From Theorem 4, we can estimate the source using the overestimated mixing matrix.

## 5   An example in EEG data analysis

The electroencephalogram (EEG) is a mixture of electrical signals coming from multiple brain sources. This is why application of ICA to EEG recently has become popular, yielding new promising results (e.g., [6]). However, compared with ICA, the sparse representation has two important advantages: 1) sources are not assumed to be mutually independent as in ICA, even be not stationary; 2) source number can be larger than the number of sensors. We believe that sparse representation is a complementary and very prospective approach in the analysis of EEG.

Here we present the results of testing the usefulness of sparse representation in the analysis of EEG data based on temporal synchronization between components. The analyzed 14-channel EEG was recorded in an experiment based on modified Sternberg memory task. Subjects were asked to memorize numbers successively presented at random positions on the computer monitor. After 2.5 s pause following by a warning signal, a "test number" was presented. If it was the same as one of the numbers in the memorized set, the subject had to press the button. This cycle, including also resting (waiting) period, was repeated 160 times (about 24 min). EEG was sampled at 256 Hz rate. Here we describe, mainly, the analysis results of one subject's data.

EEG was filtered off-line in $1-70$ Hz range, trials with artifacts were rejected by visual inspection, and a data set including 20 trials with correct response, and 20 trials with incorrect response, was selected for analysis (1 trial=2176 points). Thus we obtain a $14 \times 87040$ dimensional data matrix, denoted by $\mathbf{X}$. Using the sparse representation algorithm proposed in this paper, we decomposed the EEG signals $\mathbf{X}$ into 20 components. Denote the $20 \times 87040$ dimensional components matrix $\mathbf{S}$, which contains 20 trials for correct response, and 20 trials for incorrect response, respectively.

At first, we calculated the correlation coefficient matrices of $\mathbf{X}$ and $\mathbf{S}$, denoted by $\mathbf{R}^x$ and $\mathbf{R}^s$, respectively. We found that $\mathbf{R}^x_{i,j} \in (0.18, 1]$ (the median of $|\mathbf{R}^x_{i,j}|$ is 0.5151). In the case of components, the correlation coefficients were considerably lower (the median of $|\mathbf{R}^s_{i,j}|$ is 0.2597). And there exist many pairs of components with small correlation coefficients, e.g., $\mathbf{R}^s_{2,11} = 0.0471, \mathbf{R}^s_{8,13} = 0.0023$, etc. Furthermore, we found that the higher order correlation coefficients of these pairs are also very small (e.g., the median of absolute value of 4th order correlation is 0.1742). We would like to emphasize that, although the independence principle was not used, many pairs of components were almost independent.

According to modern brain theories, dynamics of synchronization of rhythmic activities in distinct neural networks plays a very important role in interactions between them. Thus, phase synchronization in a pair of two almost independent components $(\mathbf{s}^i_1, \mathbf{s}^i_{14})$ ($Rs_{1,14} = 0.0085$, fourth correlation coefficient 0.0026) was analyzed using method described in [7]. The synchronization index is defined by $SI(f,t) = max(SPLV(f,t) - Ssur, 0)$, where $SPLV(f,t)$ is a single-trial phase locking value at the frequency $f$ and time $t$, which has been smoothed by a window with a length of 99, and $Ssur$ is the 0.95 percentile of the distribution of 200 surrogates (the 200 pairs of surrogate data are Gaussian distributed).

Fig. 2 shows phase synchrony analysis results. The phase synchrony is observed mainly in low frequency band (1 Hz-15 Hz) and demonstrated a tendency for task-related variations.Though only ten trials are presented among the 40 trials due to page space, 32 of 40 trials shows similar characteristics.

In Fig. 3 (a), two averaged synchronization index curves are presented, which are obtained by averaging synchronization index $SI$ in the range 1-15 Hz and across 20 trials, separately for correct and incorrect response. Note the time variations of the averaged synchronization index and its higher values for correct responses, especially in the beginning and the end of the trial (preparation and response periods). To test the significance of the time and correctness effects, the synchronization index was averaged again for each 128 time points (0.5 s) for removing artificial correlation between neighboring points and submitted to Friedman nonparametric ANOVA. The test showed significance of time (p=0.013) and correctness (p=0.0017) effects. Thus, the phase synchronization between the two analyzed components was sensitive both to changes in brain activity induced by time-varying task demands and to correctness-related variations in the brain state. The higher synchronization for correct responses could be related to higher integration of brain systems required for effective information processing. This kind of phenomena also has been seen in the same analysis of EEG data from another subject (Fig. 3 (b)).

A substantial part of synchronization between raw EEG channels can be explained by volume conduction effects. Large cortical areas may work as stable unified oscillating systems, and this may account for other large part of synchronization in raw EEG. This kind of strong synchronization may make invisible synchronization appearing for brief periods, which is of special interest in brain research. To study temporally appearing synchronization, components related to the activity of more or less unified brain sources should be separated from EEG. Our first results of application of sparse representation to real EEG data support that they can help us to reveal brief periods of synchronization between brain "sources".

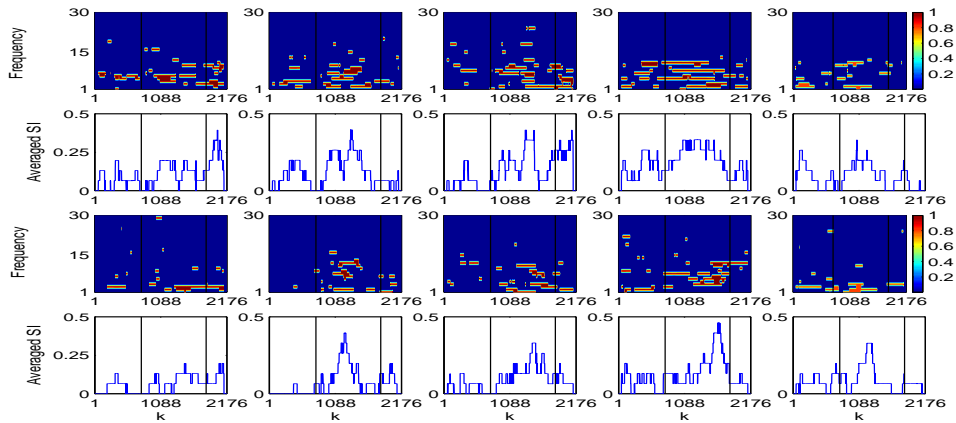

Figure 2: Time course of EEG synchrony in single trials. 1st row: time-frequency charts for 5 single trials with correct response. Synchronization index values are shown for every frequency and time sample point $(f, k)$. 2nd row: mean synchronization index averaged across frequencies in range 1-15 Hz, for the same trials as in the 1st row. 3d and 4th rows: same for five trials with incorrect response. In each subplot, the first line refers to the beginning of presentation of numbers to be memorized, the second line refers to the end of test number.

# 6  Concluding remarks

Sparse representation of data matrices and its application to blind source separation were analyzed based on a two-step approach presented in [3] in this paper. The $l^1$ norm is used

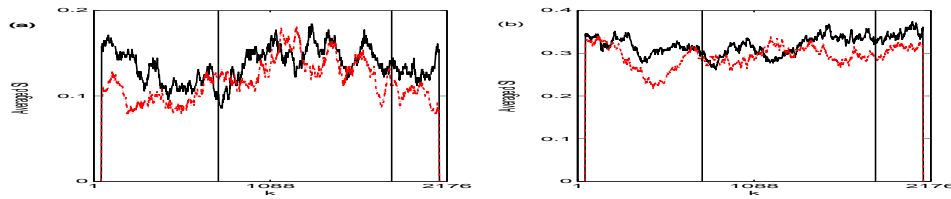

Figure 3: Time course of EEG synchrony, averaged across trials. Left: same subject as in previous figure; right: another subject. The curves show mean values of synchronization index averaged in the range 1-15 Hz and across 20 trials. Black curves are for trials with correct response, red dotted curves refers to trials with incorrect response. Solid vertical lines: as in the previous figure.

as a sparsity measure, whereas, the $l^0$ norm sparsity measure is considered for comparison and recoverability analysis of BSS. From equivalence analysis of the $l^1$ norm solution and $l^0$ norm solution presented in this paper, it is evident that if a data vector (observed vector) is generated from a sufficiently sparse source vector, then, with high probability, the $l^1$ norm solution is equal to the $l^0$ norm solution, the former in turn is equal to the source vector, which can be used for recoverability analysis of blind sparse source separation. This kind of construct that employs sparse representation can be used in BSS as in [3], especially in cases in which fewer sensors exist than sources while the source number is unknown, and sources are not completely independent. Lastly, an application example for analysis of phase synchrony in real EEG data supports its validity and performance of the proposed approach. Since the components separated by sparse representation are not constrained by the condition of complete independence, they can be used in the analysis of brain synchrony maybe more effectively than components separated by general ICA algorithms based on independence principle.

## References

[1] Chen, S., Donoho, D.L. & Saunders M. A. (1998) Automic decomposition by basis pursuit. *SIAM Journal on Scientific Computing* **20**(1):33-61.

[2] Olshausen, B.A., Sallee, P. & Lewicki, M.S. (2001) Learning sparse image codes using a wavelet pyramid architecture. *Advances in Neural Information Processing Systems 13*, pp. 887-893. Cambridge, MA: MIT Press.

[3] Zibulevsky M., Pearlmutter B. A., Boll P., & Kisilev P. (2000) Blind source separation by sparse decomposition in a signal dictionary. In Roberts, S. J. and Everson, R. M. (Eds.), *Independent Components Analysis: Principles and Practice*, Cambridge University Press.

[4] Lee, T.W., Lewicki, M.S., Girolami, M. & Sejnowski, T.J. (1999) Blind source separation of more sources than mixtures using overcomplete representations. *IEEE Signal Processing Letter* **6**(4):87-90.

[5] Donoho, D.L. & Elad, M. (2003) Maximal sparsity representation via $l^1$ minimization. *the Proc. Nat. Aca. Sci.* **100**:2197-2202.

[6] Makeig, S., Westerfi eld, M., Jung, T.P., Enghoff, S., Townsend, J., Courchesne, E. & Sejnowski, T.J. (2002) Dynamic brain sources of visual evoked responses. *Science* **295**:690-694.

[7] Le Van Quyen, M., Foucher, J., Lachaux, J.P., Rodriguez, E., Lutz, A., Martinerie, J. & Varela, F.J. (2001) Comparison of Hilbert transform and wavelet methods for the analysis of neuronal synchrony. *Journal of Neuroscience Methods* **111**:83-98.
